# Augmented Functional Time Series Representation and Forecasting with Gaussian Processes

**Nicolas Chapados and Yoshua Bengio**
Department of Computer Science and Operations Research
University of Montréal
Montréal, Québec, Canada H3C 3J7
{chapados,bengioy}@iro.umontreal.ca

## Abstract

We introduce a functional representation of time series which allows forecasts to be performed over an unspecified horizon with progressively-revealed information sets. By virtue of using Gaussian processes, a complete covariance matrix between forecasts at several time-steps is available. This information is put to use in an application to actively trade price spreads between commodity futures contracts. The approach delivers impressive out-of-sample risk-adjusted returns after transaction costs on a portfolio of 30 spreads.

## 1 Introduction

Classical time-series forecasting models, such as ARMA models [6], assume that forecasting is performed at a fixed horizon, which is implicit in the model. An overlaying deterministic time trend may be fit to the data, but is generally of fixed and relatively simple functional form (e.g. linear, quadratic, or sinusoidal for periodic data). To forecast beyond the fixed horizon, it is necessary to iterate forecasts in a multi-step fashion. These models are good at representing the short-term dynamics of the time series, but degrade rapidly when longer-term forecasts must be made, usually quickly converging to the unconditional expectation of the process after removal of the deterministic time trend. This is a major issue in applications that require a forecast over a **complete future trajectory**, and not a single (or restricted) horizon. These models are also constrained to deal with regularly-sampled data, and make it difficult to condition the time trend on explanatory variables, especially when iteration of short-term forecasts has to be performed. To a large extent, the same problems are present with non-linear generalizations of such models, such as time-delay or recurrent neural networks [1], which simply allow the short-term dynamics to become nonlinear but leave open the question of forecasting complete future trajectories.

*Functional Data Analysis* (FDA) [10] has been proposed in the statistical literature as an answer to some of these concerns. The central idea is to consider a whole curve as an example (specified by a finite number of samples $\langle t, y_t \rangle$), which can be represented by coefficients in a non-parametric basis expansion such as splines. This implies learning about complete trajectories as a function of time, hence the "functional" designation. Since time is viewed as an independent variable, the approach can forecast at arbitrary horizons and handle irregularly-sampled data. Typically, FDA is used without explanatory time-dependent variables, which are important for the kind of applications we shall be considering. Furthermore, the question remains of how to integrate a progressively-revealed information set in order to make increasingly more precise forecasts of the same future trajectory. To incorporate conditioning information, we consider here the output of a prediction to be a whole forecasting curve (as a function of $t$).

The motivation for this work comes from forecasting and actively trading price spreads between commodity futures contracts (see, e.g., [7], for an introduction). Since futures contracts expire and have a finite duration, this problem is characterized by the presence of a large number of separate

historical time series, which all can be of relevance in forecasting a new time series. For example, we expect seasonalities to affect similarly all the series. Furthermore, conditioning information, in the form of macroeconomic variables, can be of importance, but exhibit the cumbersome property of being released periodically, with explanatory power that varies across the forecasting horizon. In other words, when making a very long-horizon forecast, the model should not incorporate conditioning information in the same way as when making a short- or medium-term forecast. A possible solution to this problem is to have multiple models for forecasting each time series, one for each time scale. However, this is hard to work with, requires a high degree of skill on the part of the modeler, and is not amenable to robust automation when one wants to process hundreds of time series. In addition, in order to measure risk associated with a particular trade (buying at time $t$ and selling at time $t'$), we need to estimate the *covariance of the price predictions* associated with these two points in the trajectory.

These considerations motivate the use of Gaussian processes, which naturally provide a covariance matrix between forecasts made at several points. To tackle the challenging task of forecasting and trading spreads between commodity futures, we introduce here a form of functional data analysis in which the function to be forecast is indexed both by the date of availability of the information set and by the forecast horizon. The predicted trajectory is thus represented as a functional object associated with a distribution, a Gaussian process, from which the risk of different trading decisions can readily be estimated. This approach allows incorporating input variables that cannot be assumed to remain constant over the forecast horizon, like statistics of the short-term dynamics.

**Previous Work**    Gaussian processes for time-series forecasting have been considered before. Multi-step forecasts are explicitly tackled by [4], wherein uncertainty about the intermediate values is formally incorporated into the predictive distribution to obtain more realistic uncertainty bounds at longer horizons. However, this approach, while well-suited to purely autoregressive processes, does not appear amenable to the explicit handling of exogenous input variables. Furthermore, it suffers from the restriction of only dealing with regularly-sampled data. Our approach is inspired by the $CO_2$ model of [11] as an example of application-specific covariance function engineering.

## 2    The Model

We consider a set of $N$ real time series each of length $M_i$, $\{y_t^i\}, i = 1, \ldots, N$ and $t = 1, \ldots, M_i$. In our application each $i$ represents a different year, and the series is the sequence of commodity spread prices during the period where it is traded. The lengths of all series are not necessarily identical, but we shall assume that the time periods spanned by the series are "comparable" (e.g. the same range of days within a year if the series follow an annual cycle) so that knowledge from past series can be transferred to a new one to be forecast. The **forecasting problem** is that given observations from the complete series $i = 1, \ldots, N-1$ and from a *partial last series*, $\{y_t^N\}, t = 1, \ldots, M_N$, we want to extrapolate the last series until a predetermined endpoint, i.e. characterize the joint distribution of $\{y_\tau^N\}, \tau = M_N + 1, \ldots, M_N + H$. We are also given a set of non-stochastic explanatory variables specific to each series, $\{\mathbf{x}_t^i\}$, where $\mathbf{x}_t^i \in \mathbb{R}^d$. Our objective is to find an effective representation of $P(\{y_\tau^N\}_{\tau = M_N+1,\ldots,M_N+H} \mid \{\mathbf{x}_t^i, y_t^i\}_{t=1,\ldots,M_i}^{i=1,\ldots,N})$, with $\tau, i$ and $t$ ranging, respectively over the forecasting horizon, the available series and the observations within a series.

**Gaussian Processes**    Assuming that we are willing to accept a normally-distributed posterior, Gaussian processes [8, 11, 14] have proved a general and flexible tool for nonlinear regression in a Bayesian framework. Given a training set of $M$ input–output pairs $\langle \mathbf{X} \in \mathbb{R}^{M \times d}, \mathbf{y} \in \mathbb{R}^M \rangle$, a set of $M'$ test point locations $\mathbf{X}_* \in \mathbb{R}^{M' \times d}$ and a positive semi-definite covariance function $k : \mathbb{R}^d \times \mathbb{R}^d \mapsto \mathbb{R}$, the joint posterior distribution of the test outputs $\mathbf{y}_*$ follows a normal with mean and covariance given by

$$\mathbb{E}\left[\mathbf{y}_* \mid \mathbf{X}, \mathbf{X}_*, \mathbf{y}\right] = K(\mathbf{X}_*, \mathbf{X})\mathbf{\Lambda}^{-1}\mathbf{y}, \tag{1}$$

$$\mathrm{Cov}\left[\mathbf{y}_* \mid \mathbf{X}, \mathbf{X}_*, \mathbf{y}\right] = K(\mathbf{X}_*, \mathbf{X}_*) - K(\mathbf{X}_*, \mathbf{X})\mathbf{\Lambda}^{-1}K(\mathbf{X}, \mathbf{X}_*), \tag{2}$$

where we have set $\mathbf{\Lambda} = K(\mathbf{X}, \mathbf{X}) + \sigma_n^2 I_M$, with $K$ the matrix of covariance evaluations, $K(\mathbf{U}, \mathbf{V})_{i,j} \triangleq k(\mathbf{U}_i, \mathbf{V}_j)$, and $\sigma_n^2$ the assumed process noise level. The specific form of the covariance function used in our application is described below, after introducing the representation used for forecasting.

**Functional Representation for Forecasting**    In the spirit of functional data analysis, a first attempt at solving the forecasting problem is to set it forth in terms of regression from the input variables to the series values, adding to the inputs an explicit time index $t$ and series identity $i$,

$$\mathbb{E}\left[y_t^i \big| \mathcal{I}_{t_0}^i\right] = f(i, t, \mathbf{x}_{t|t_0}^i) \qquad \mathrm{Cov}\left[y_t^i, y_{t'}^{i'} \big| \mathcal{I}_{t_0}^i\right] = g(i, t, \mathbf{x}_{t|t_0}^i, i', t', \mathbf{x}_{t'|t_0}^{i'}), \qquad (3)$$

these expressions being conditioned on the *information set* $\mathcal{I}_{t_0}^i$ containing information up to time $t_0$ of series $i$ (we assume that all prior series $i' < i$ are also included in their entirety in $\mathcal{I}_{t_0}^i$). The notation $\mathbf{x}_{t|t_0}^i$ denotes a forecast of $\mathbf{x}_t^i$ given information available at $t_0$. Functions $f$ and $g$ result from Gaussian process training, eq. (1) and (2), using information in $\mathcal{I}_{t_0}^i$. To extrapolate over the unknown horizon, one simply evaluates $f$ and $g$ with the series identity index $i$ set to $N$ and the time index $t$ within a series ranging over the elements of $\tau$ (forecasting period). Owing to the smoothness properties of an adequate covariance function, one can expect the last time series (whose starting portion is present in the training data) to be smoothly extended, with the Gaussian process borrowing from prior series, $i < N$, to guide the extrapolation as the time index reaches far enough beyond the available data in the last series.

The principal difficulty with this method resides in handling the exogenous inputs $\mathbf{x}_{t|t_0}^N$ over the forecasting period: the realizations of these variables, $\mathbf{x}_t^N$, are not usually known at the time the forecast is made and must be extrapolated with some reasonableness. For slow-moving variables that represent a "level" (as opposed to a "difference" or a "return"), one can conceivably keep their value constant to the last known realization across the forecasting period. However, this solution is restrictive, problem-dependent, and precludes the incorporation of short-term dynamics variables (e.g. the first differences over the last few time-steps) if desired.

**Augmenting the Functional Representation**    We propose in this paper to augment the functional representation with an additional input variable that expresses the time *at which* the forecast is being made, in addition to the time *for which* the forecast is made. We shall denote the former the *operation time* and the latter the *target time*. The distinction is as follows: **operation time** represents the time at which the other input variables are observed and the time at which, conceptually, a forecast of the entire future trajectory is performed. In contrast, **target time** represents time at a point of the predicted target series (beyond operation time), given the information known at the operation time.

As previously, the time series index $i$ remains part of the inputs. In this framework, forecasting is performed by holding the time series index constant to $N$, the operation time constant to the time $M_N$ of the last observation, the other input variables constant to their last-observed values $\mathbf{x}_{M_N}^N$, and *varying the target time* over the forecasting period $\tau$. Since we are not attempting to extrapolate the inputs beyond their intended range of validity, this approach admits general input variables, without restriction as to their type, and whether they themselves can be forecast.

It can be convenient to represent the target time as a positive delta $\Delta$ from the operation time $t_0$. In contrast to eq. (3), this yields the representation

$$\mathbb{E}\left[y_{t_0+\Delta}^i \big| \mathcal{I}_{t_0}^i\right] = f(i, t_0, \Delta, \mathbf{x}_{t_0}^i) \qquad \mathrm{Cov}\left[y_{t_0+\Delta}^i, y_{t_0'+\Delta'}^{i'} \big| \mathcal{I}_{t_0}^i\right] = g(i, t_0, \Delta, \mathbf{x}_{t_0}^i, i', t_0', \Delta', \mathbf{x}_{t_0'}^{i'}),$$
$$(4)$$

where we have assumed the operation time to coincide with the end of the information set. Note that this augmentation allows to dispense with the problematic extrapolation $\mathbf{x}_{t|t_0}^i$ of the inputs, instead allowing a direct use of the last available values $\mathbf{x}_{t_0}^i$. Moreover, from a given information set, nothing precludes forecasting the same trajectory from several operation times $t' < t_0$, which can be used as a means of evaluating the stability of the obtained forecast.

The obvious downside to augmentation lies in the greater computational cost it entails. In particular, the training set must contain sufficient information to represent the output variable for *many combinations of operation and target times* that can be provided as input. In the worst case, this implies that the number of training examples grows quadratically with the length of the training time series. In practice, a downsampling scheme is used wherein only a fixed number of target-time points is sampled for every operation-time point.[1]

**Covariance Function** We used a modified form of the *rational quadratic* covariance function with hyperparameters for automatic relevance determination [11], which is expressed as

$$k_{\text{AUG-RQ}}(\mathbf{u}, \mathbf{v}; \ell, \alpha, \sigma_f, \sigma_{\text{TS}}) = \sigma_f^2 \left(1 + \frac{1}{2\alpha} \sum_{k=1}^{d} \frac{(\mathbf{u}_k - \mathbf{v}_k)^2}{\ell_k^2}\right)^{-\alpha} + \sigma_{\text{TS}}^2 \delta_{i_\mathbf{u}, i_\mathbf{v}}, \qquad (5)$$

where $\delta_{j,k} \overset{\triangle}{=} I[j = k]$ is the Kronecker delta. The variables $\mathbf{u}$ and $\mathbf{v}$ are values in the augmented representation introduced previously, containing the three variables representing time (current time-series index or year, operation time, target time) as well as the additional explanatory variables. The notation $i_\mathbf{u}$ denotes the time-series index component $i$ of input variable $\mathbf{u}$. The last term of the covariance function, the Kronecker delta, is used to induce an increased similarity among points that belong to the same time series (e.g. the same spread trading year). By allowing a series-specific average level to be maintained into the extrapolated portion, the presence of this term was found to bring better forecasting performance. The hyperparameters $\ell_i, \alpha, \sigma_f, \sigma_{\text{TS}}, \sigma_n$ are found by maximizing the marginal likelihood on the training set by a standard conjugate gradient optimization [11]. For tractability, we rely on a two-stage training procedure, wherein hyperparameter optimization is performed on a fairly small training set ($M = 500$) and final training is done on a larger set ($M = 2250$), keeping hyperparameters fixed.

## 3 Evaluating Forecasting Performance

To establish the benefits of the proposed functional representation for forecasting commodity spread prices, we compared it against other likely models on three common grain and grain-related spreads:[2] the January–July Soybeans, May–September Soybean Meal, and March–July Chicago Hard Red Wheat. The forecasting task is to *predict the complete future trajectory* of each spread (taken individually), from 200 days before maturity until maturity.

**Methodology** Realized prices in the previous trading years are provided from 250 days to maturity, using data going back to 1989. The first test year is 1994. Within a given trading year, the time variables represent the number of calendar days to maturity of the near leg; since no data is observed on week-ends, training examples are sampled on an irregular time scale. Performance evaluation proceeds through a *sequential validation* procedure [2]: within a trading year, we first train models 200 days before maturity and obtain a first forecast for the future price trajectory. We then retrain models every 25 days, and obtain revised portions of the remainder of the trajectory. Proceeding sequentially, this operation is repeated for succeeding trading years. All forecasts are compared amongst models on squared-error and negative log-likelihood criteria (see "assessing significance", below). Input variables are subject to minimal preprocessing: we standardize them to zero mean and unit standard deviation. The price targets require additional treatment: since the price level of a spread can vary significantly from year to year, we normalize the price trajectories to *start at zero* at the start of every trading year, by subtracting the first price. Furthermore, in order to get slightly better behaved optimization, we divide the price targets by their overall standard deviation.

**Models Compared** The "complete" model to be compared against others is based on the augmented-input representation Gaussian process with the modified rational quadratic covariance function eq. (5). In addition to the three variables required for the representation of time, the following inputs were provided to the model: (i) the current spread price and the price of the three nearest futures contracts on the underlying commodity term structure, (ii) economic variables (the stock-to-use ratio and year-over-year difference in total ending stocks) provided on the underlying commodity by the U.S. Department of Agriculture [13]. This model is denoted **AugRQ/all-inp**. An example of the sequence of forecasts made by this model, repeated every 25 times steps, is shown in the upper panel of Figure 1.

To determine the value added by each type of input variable, we include in the comparison two models based on exactly on the same architecture, but providing less inputs: **AugRQ/less-inp** does

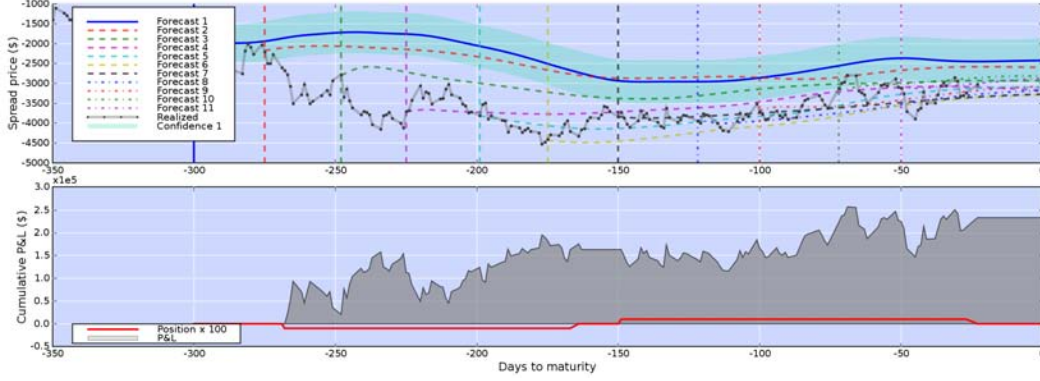

Figure 1: *Top Panel: Illustration of multiple forecasts, repeated every 25 days, of the 1996 March–July Wheat spread (dashed lines); realized price is in gray. Although the first forecast (smooth solid blue, with confidence bands) mistakes the overall price level, it approximately correctly identifies local price maxima and minima, which is sufficient for trading purposes. Bottom Panel: Position taken by the trading model (in red: short, then neutral, then long), and cumulative profit of that trade (gray).*

not include the economic variables. **AugRQ/no-inp** further removes the price inputs, leaving only the time-representation inputs. Moreover, to quantify the performance gain of the augmented representation of time, the model **StdRQ/no-inp** implements a "standard time representation" that would likely be used in a functional data analysis model; as described in eq. (3), this uses a single time variable instead of splitting the representation of time between the operation and target times.

Finally, we compare against simpler models: **Linear/all-inp** uses a dot-product covariance function to implement Bayesian linear regression, using the full set of input variables described above. And **AR(1)** is a simple linear autoregressive model. The predictive mean and covariance matrix for this last model are established as follows (see, e.g. [6]). We consider the scalar data generating process

$$y_t = \phi\, y_{t-1} + \varepsilon_t, \qquad \varepsilon_t \overset{\text{iid}}{\sim} \mathcal{N}(0, \sigma^2),\tag{6}$$

where the process $\{y_t\}$ has an unconditional mean of zero.[3] Given information available at time $t$, $\mathcal{I}_t$, the $h$-step ahead forecast from time $t$ under this model, has conditional expectation and covariance (with the $h'$-step ahead forecast), expressed as

$$\mathbb{E}\left[y_{t+h}\,|\,\mathcal{I}_t\right] = \phi^h y_t, \qquad \text{Cov}\left[y_{t+h|t}, y_{t+h'|t}\,|\,\mathcal{I}_t\right] = \sigma^2 \phi^{h+h'} \frac{1 - \phi^{-2\min(h,h')}}{\phi^2 - 1}.$$

**Assessing Significance of Forecasting Performance Differences**   For each trajectory forecast, we measure the squared error (SE) made at each time-step along with the negative log-likelihood (NLL) of the realized price under the predictive distribution. To account for differences in target variable distribution throughout the years, we normalize the SE by dividing it by the standard deviation of the test targets in a given year. Similarly, we normalize the NLL by subtracting the likelihood of a univariate Gaussian distribution estimated on the test targets of the year.

Due to the serial correlation it exhibits, the time series of performance differences (either SE or NLL) between two models cannot directly be subjected to a standard $t$-test of the null hypothesis of no difference in forecasting performance. The well-known Diebold-Mariano test [3] corrects for this correlation structure in the case where a *single time series* of performance differences is available. This test is usually expressed as follows. Let $\{d_t\}$ be the sequence of *error differences* between two models to be compared. Let $\bar{d} = \frac{1}{M}\sum_t d_t$ be the mean difference. The sample variance of $\bar{d}$ is readily shown [3] to be

$$\hat{v}_{\text{DM}} \overset{\triangle}{=} \text{Var}[\bar{d}] = \frac{1}{M}\sum_{k=-K}^{K} \hat{\gamma}_k,$$

Table 1: *Forecast performance difference between **AugRQ/all-inp** and all other models, for the three spreads studied. For both the Squared Error and NLL criteria, the value of the cross-correlation-corrected statistic is listed (CCC) along with its p-value under the null hypothesis. A negative CCC statistic indicates that **AugRQ/all-inp** beats the other model on average.*

| | Soybeans 1–7 | | | | Soybean Meal 5–9 | | | | Wheat 3–7 | | | |
| | Sq. Error | | NLL | | Sq. Error | | NLL | | Sq. Error | | NLL | |
| | CCC | $p$ | CCC | $p$ | CCC | $p$ | CCC | $p$ | CCC | $p$ | CCC | $p$ |
|---|---|---|---|---|---|---|---|---|---|---|---|---|
| AugRQ/less-inp | $-0.86$ | $0.39$ | $-0.89$ | $0.37$ | $-1.05$ | $0.29$ | $-0.95$ | $0.34$ | $-0.05$ | $0.96$ | $1.06$ | $0.29$ |
| AugRQ/no-inp | $-1.68$ | $0.09$ | $-1.73$ | $0.08$ | $-1.78$ | $0.08$ | $-2.42$ | $0.02$ | $-2.75$ | $0.01$ | $-2.42$ | $0.02$ |
| Linear/all-inp | $-1.53$ | $0.13$ | $-1.33$ | $0.18$ | $-1.61$ | $0.11$ | $-2.00$ | $0.05$ | $-4.20$ | $10^{-4}$ | $-3.45$ | $10^{-3}$ |
| $AR(1)$ | $-4.24$ | $10^{-5}$ | $-0.44$ | $0.66$ | $-2.53$ | $0.01$ | $0.12$ | $0.90$ | $-6.50$ | $0.00$ | $-6.07$ | $10^{-9}$ |
| StdRQ/no-inp | $-2.44$ | $0.01$ | $-1.04$ | $0.30$ | $-2.69$ | $0.01$ | $-1.08$ | $0.28$ | $-2.67$ | $0.01$ | $-9.36$ | $0.00$ |

where $M$ is the sequence length and $\hat{\gamma}_k$ is an estimator of the lag-$k$ autocovariance of the $d_t$s. The maximum lag order $K$ is a parameter of the test and must be determined empirically. Then the statistic $DM = \bar{d}/\sqrt{\hat{v}_{DM}}$ is asymptotically distributed as $\mathcal{N}(0,1)$ and a classical test of the null hypothesis $\bar{d} = 0$ can be performed.

Unfortunately, even the Diebold-Mariano correction for autocorrelation is not sufficient to compare models in the present case. Due to the repeated forecasts made for the same time-step across *several iterations of sequential validation*, the error sequences are likely to be *cross-correlated* since they result from models estimated on strongly overlapping training sets. This suggests that an additional correction should be applied to account for this cross-correlation across test sets, expressed as

$$\hat{v}_{\text{CCC}-\text{DM}} = \frac{1}{M^2}\left(\sum_i M_i \sum_{k=-K}^{K} \hat{\gamma}_k^i + \sum_i \sum_{j\neq i} M_{i\cap j} \sum_{k=-K'}^{K'} \hat{\gamma}_k^{i,j}\right), \qquad (7)$$

where $M_i$ is the number of examples in test set $i$, $M = \sum_i M_i$ is the total number of examples, $M_{i\cap j}$ is the number of time-steps where test sets $i$ and $j$ overlap, $\hat{\gamma}_k^i$ denote the estimated lag-$k$ autocovariances within test set $i$, and $\hat{\gamma}_k^{i,j}$ denote the estimated lag-$k$ cross-covariances between test sets $i$ and $j$. The maximum lag order for cross-covariances, $K'$, is possibly different from $K$ (our experiments used $K = K' = 15$). This revised variance estimator was used in place of the usual Diebold-Mariano statistic in the results presented below.

**Results**  Results of the forecasting *performance difference* between **AugRQ/all-inp** and all other models is shown in Table 1. We observe that **AugRQ/all-inp** generally beats the others on both the SE and NLL criteria, often statistically significantly so. In particular, the augmented representation of time is shown to be of value (i.e. comparing against **StdRQ/no-inp**). Moreover, the Gaussian process is capable of making good use of the additional price and economic input variables, although not always with the traditionally accepted levels of significance.

## 4   Application: Trading a Portfolio of Spreads

We applied this forecasting methodology based on an augmented representation of time to trading a portfolio of spreads. Within a given trading year, we apply an information-ratio criterion to greedily determine the best trade into which to enter, based on the entire price forecast (until the end of the year) produced by the Gaussian process. More specifically, let $\{p_t\}$ be the future prices forecast by the model at some operation time (presumably the time of last available element in the training set). The expected forecast dollar profit of buying at $t_1$ and selling at $t_2$ is simply given by $p_{t_2} - p_{t_1}$. Of course, a prudent investor would take trade risk into consideration. A simple approximation of risk is given by the trade profit volatility. This yields the *forecast information ratio*[4] of the trade

$$\widehat{IR}(t_1, t_2) = \frac{\mathbb{E}[p_{t_2} - p_{t_1}|\mathcal{I}_{t_0}]}{\sqrt{\text{Var}[p_{t_2} - p_{t_1}|\mathcal{I}_{t_0}]}}, \qquad (8)$$

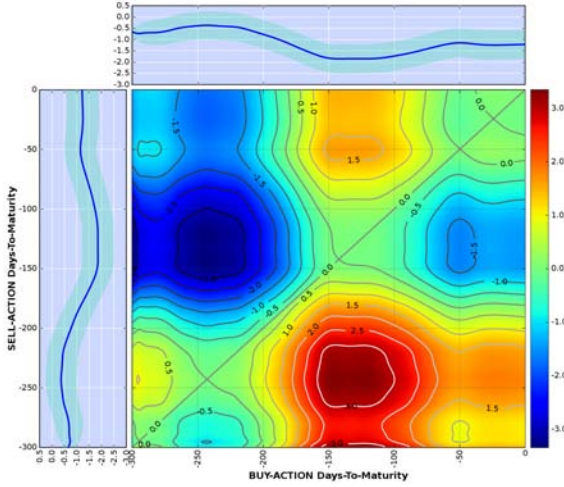

Figure 2: *After a price trajectory forecast (in the top and left portions of the figure), all possible pairs of buy-day/sell-day are evaluated on a trade information ratio criterion, whose results are shown by the level plot. The best trade is selected, here shorting 235 days before maturity with forecast price at a local maximum, and covering 100 days later at a local minimum.*

Table 2: *Financial performance statistics for the 30-spread portfolio on the 1994–2007 (until April 30) period, and two disjoint sub-periods. All returns are expressed in excess of the risk-free rate. The information ratio statistics are annualized. Skewness and excess kurtosis are on the monthly return distributions. Drawdown duration is expressed in calendar days. The model displays good performance for moderate risk.*

|  | Full Period | 1994/01 2002/12 | 2003/01 2007/04 |
|---|---|---|---|
| Avg Annual Return | 7.3% | 5.9% | 10.1% |
| Avg Annual Stddev | 4.1% | 4.0% | 4.1% |
| Information Ratio | 1.77 | 1.45 | 2.44 |
| Skewness | 0.68 | 0.65 | 0.76 |
| Excess Kurtosis | 3.40 | 4.60 | 1.26 |
| Best Month | 6.0% | 6.0% | 4.8% |
| Worst Month | −3.4% | −3.4% | −1.8% |
| Percent Months Up | 71% | 67% | 77% |
| Max. Drawdown | −7.7% | −7.7% | −4.0% |
| Drawdown Duration | 653 | 653 | 23 |
| Drawdown From | 1997/02 | 1997/02 | 2004/06 |
| Drawdown Until | 1998/11 | 1998/11 | 2004/07 |

where $\mathrm{Var}[p_{t_2} - p_{t_1}|\mathcal{I}_{t_0}]$ can be computed as $\mathrm{Var}[p_{t_1}|\mathcal{I}_{t_0}] + \mathrm{Var}[p_{t_1}|\mathcal{I}_{t_0}] - 2\,\mathrm{Cov}[p_{t_1}, p_{t_2}|\mathcal{I}_{t_0}]$, each quantity being separately obtainable from the Gaussian process forecast, *cf.* eq. (2). The trade decision is made in one of two ways, depending on whether a position has already been opened: (i) When making a decision at time $t_0$, if a position has *not yet been entered* for the spread in a given trading year, eq. (8) is maximized with respect to unconstrained $t_1, t_2 \geq t_0$. An illustration of this criterion is given in Figure 2, which corresponds to the first decision made when trading the spread shown in Figure 1. (ii) In contrast, if a position *has already been opened*, eq. (8) is only maximized with respect to $t_2$, keeping $t_1$ fixed at $t_0$. This corresponds to revising the exit point of an existing position. Simple additional filters are used to avoid entering marginal trades: we impose a trade duration of at least four days, a minimum forecast IR of 0.25 and a forecast standard deviation of the price sequence of at least 0.075. These thresholds have not been tuned extensively; they were used only to avoid trading on an approximately flat price forecast.

We applied these ideas to trading a portfolio of 30 spreads, selected among the following commodities: Cotton (2 spreads), Feeder Cattle (2), Gasoline (1), Lean Hogs (7), Live Cattle (1), Natural Gas (2), Soybean Meal (5), Soybeans (5), Wheat (5). The spreads were selected on the basis of their good performance on the 1994–2002 period. Our simulations were carried on the 1994–2007 period, using historical data (for Gaussian process training) dating back to 1989. Transaction costs were assumed to be 5 basis points per spread leg traded. Spreads were never traded later than 25 calendar days before maturity of the near leg. Relative returns are computed using as a notional amount half the total exposure incurred by both legs of the spread.[5] Financial performance results on the complete test period and two disjoint sub-periods (which correspond, until end-2002 to the model selection period, and after 2003 to a true out-of-sample evaluation) are shown in Table 2. In all sub-periods, but particularly since 2003, the portfolio exhibits a very favorable risk-return profile, including positive skewness and acceptable excess kurtosis.[6] A plot of cumulative returns, number of open positions and monthly returns appears in Figure 3.

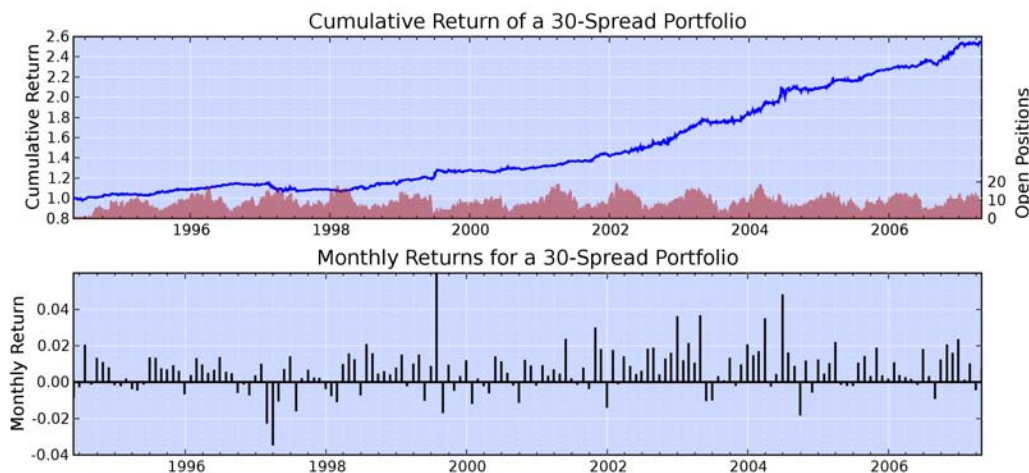

Figure 3: ***Top Panel:*** *cumulative excess return after transaction costs of a portfolio of 30 spreads traded according to the maximum information-ratio criterion; the bottom part plots the number of positions open at a time (right axis).* ***Bottom Panel:*** *monthly portfolio relative excess returns; we observe the significant positive skewness in the distribution.*

## 5   Future Work and Conclusions

We introduced a flexible functional representation of time series, capable of making long-term forecasts from progressively-revealed information sets and of handling multiple irregularly-sampled series as training examples. We demonstrated the approach on a challenging commodity spread trading application, making use of a Gaussian process' ability to compute a complete covariance matrix between several test outputs. Future work includes making more systematic use of approximation methods for Gaussian processes (see [9] for a survey). The specific usage pattern of the Gaussian process may guide the approximation: in particular, since we know in advance the test inputs, the problem is intrinsically one of *transduction*, and the Bayesian Committee Machine [12] could prove beneficial.

## Footnotes

[1]This number was 15 in our experiments, and these were not regularly spaced, with longer horizons spaced farther apart. Furthermore, the original daily frequency of the data was reduced to keep approximately one operation-time point per week.

[2]Our convention is to first give the *short leg* of the spread, followed by the *long leg*. Hence, Soybeans 1–7 should be interpreted as taking a short position (i.e. selling) in the January Soybeans contract and taking an offsetting long (i.e. buying) in the July contract. Traditionally, intra-commodity spread positions are taken so as to match the number of contracts on both legs — the number of short contracts equals the number of long ones — not the dollar value of the long and short sides.

[3]In our experiments, we estimate an independent empirical mean for each trading year, which is subtracted from the prices before proceeding with the analysis.

[4]An *information ratio* is defined as the average return of a portfolio in excess of a benchmark, divided by the standard deviation of the excess return distribution; see [5] for more details.

[5]This is a conservative assumption, since most exchanges impose considerably reduced margin requirements on recognized spreads.

[6]By way of comparison, over the period 1 Jan. 1994–30 Apr. 2007, the S&P 500 index has an information ratio of approximately 0.37 against the U.S. three-month treasury bills.

## References

[1] C. Bishop. *Neural Networks for Pattern Recognition*. Oxford University Press, 1995.

[2] N. Chapados and Y. Bengio. Cost functions and model combination for VaR-based asset allocation using neural networks. *IEEE Transactions on Neural Networks*, 12(4):890–906, July 2001.

[3] F. X. Diebold and R. S. Mariano. Comparing predictive accuracy. *Journal of Business & Economic Statistics*, 13(3):253–263, July 1995.

[4] A. Girard, C. E. Rasmussen, J. Q. Candela, and R. Murray-Smith. Gaussian process priors with uncertain inputs – application to multiple-step ahead time series forecasting. In S. T. S. Becker and K. Obermayer, editors, *Advances in Neural Information Processing Systems 15*, pages 529–536. MIT Press, 2003.

[5] R. C. Grinold and R. N. Kahn. *Active Portfolio Management*. McGraw Hill, 1999.

[6] J. D. Hamilton. *Time Series Analysis*. Princeton University Press, 1994.

[7] J. C. Hull. *Options, Futures and Other Derivatives*. Prentice Hall, Englewood Cliffs, NJ, sixth edition, 2005.

[8] A. O'Hagan. Curve fitting and optimal design for prediction. *Journal of the Royal Statistical Society B*, 40:1–42, 1978. (With discussion).

[9] J. Quionero-Candela and C. E. Rasmussen. A unifying view of sparse approximate gaussian process regression. *Journal of Machine Learning Research*, 6:1939–1959, 2005.

[10] J. O. Ramsay and B. W. Silverman. *Functional Data Analysis*. Springer, second edition, 2005.

[11] C. E. Rasmussen and C. K. I. Williams. *Gaussian Processes for Machine Learning*. MIT Press, 2006.

[12] V. Tresp. A bayesian committee machine. *Neural Computation*, 12:2719–2741, 2000.

[13] U.S. Department of Agriculture. Economic research service data sets. WWW publication. Available at http://www.ers.usda.gov/Data/.

[14] C. K. I. Williams and C. E. Rasmussen. Gaussian processes for regression. In D. S. Touretzky, M. C. Mozer, and M. E. Hasselmo, editors, *Advances in Neural Information Processing Systems 8*, pages 514–520. MIT Press, 1996.

